# Kernel Embeddings of Latent Tree Graphical Models

**Le Song**
College of Computing
Georgia Institute of Technology
lsong@cc.gatech.edu

**Ankur P. Parikh**
School of Computer Science
Carnegie Mellon University
apparikh@cs.cmu.edu

**Eric P. Xing**
School of Computer Science
Carnegie Mellon University
epxing@cs.cmu.edu

## Abstract

Latent tree graphical models are natural tools for expressing long range and hierarchical dependencies among many variables which are common in computer vision, bioinformatics and natural language processing problems. However, existing models are largely restricted to discrete and Gaussian variables due to computational constraints; furthermore, algorithms for estimating the latent tree structure and learning the model parameters are largely restricted to heuristic local search. We present a method based on kernel embeddings of distributions for latent tree graphical models with continuous and non-Gaussian variables. Our method can recover the latent tree structures with provable guarantees and perform local-minimum free parameter learning and efficient inference. Experiments on simulated and real data show the advantage of our proposed approach.

## 1 Introduction

Real world problems often produce high dimensional features with sophisticated statistical dependency structures. One way to compactly model these statistical structures is to use probabilistic graphical models that relate the observed features to a set of latent or hidden variables. By defining a joint probabilistic model over observed and latent variables, the marginal distribution of the observed variables is obtained by integrating out the latent ones. This allows complex distributions over observed variables (*e.g.*, clique models) to be expressed in terms of more tractable joint models (*e.g.*, tree models) over the augmented variable space. Probabilistic models with latent variables have been deployed successfully to a diverse range of problems such as in document analysis [3], social network modeling [10], speech recognition [18] and bioinformatics [5].

In this paper, we will focus on latent variable models where the latent structures are trees (we call it a "latent tree" for short). In these tree-shaped graphical models, the leaves are the set of observed variables (*e.g.*, taxa, pixels, words) while the internal nodes are hidden and intuitively "represent" the common properties of their descendants (*e.g.*, distinct ancestral species, objects in an image, latent semantics). This class of models strike a nice balance between their representation power (*e.g.*, ability to model cliques) and the complexity of learning and inference processes on these structures (*e.g.*, message passing is exact on trees). In particular, we will study the problems of estimating the latent tree structures, learning the model parameters and performing inference on these models for *continuous and non-Gaussian* variables where it is not easy to specify a parametric family.

In previous works, the challenging problem of estimating the structure of latent trees has largely been tackled by heuristics since the search space of structures is intractable. For instance, Zhang et al. [28] proposed a search heuristic for hierarchical latent class models by defining a series of local search operations and using EM to compute the likelihood of candidate structures. Harmeling and Williams [8] proposed a greedy algorithm to learn binary trees by joining two nodes with a high mutual information and iteratively performing EM to compute the mutual information among newly added hidden nodes. Alternatively, Bayesian hierarchical clustering [9] is an agglomerative clustering technique that merges clusters based on a statistical hypothesis test. Many other local search heuristics based on maximum parsimony and maximum likelihood methods can also be found from

the phylogenetic community [21]. However, none of these methods extend easily to the nonparametric case since they require the data to be discrete or to have a parametric form such that statistical tests or likelihoods/EM can be easily computed.

Given the structures of the latent trees, learning the model parameters has predominantly relied on likelihood maximization and local search heuristics such as expectation maximization (EM) [6]. Besides the problem of local minima, non-Gaussian statistical features such as multimodality and skewness may pose additional challenges for EM. For instance, parametric models such as mixture of Gaussians may lead to an exponential blowup in terms of representation during the inference stage of EM, so further approximations may be needed to make these cases tractable. Furthermore, EM can require many iterations to reach a prescribed training precision.

In this paper, we propose a method for latent tree models with continuous and non-Gaussian observation based on the concept of kernel embedding of distributions [23]. The problems we try to address are: how to estimate the structures of latent trees with provable guarantees, and how to perform local-minimum-free parameter learning and efficient inference given the tree structures, all in nonparametric fashion. The main flavor of our method is to exploit the spectral properties of the joint embedding (or covariance operators) in both the structure recovery and learning/inference stage. For the former, we define a distance measure between variables based on the singular value decomposition of covariance operators. This allows us to generalize some of the distance based latent tree learning procedures such as neighbor joining [20] and the recursive grouping methods [4] to the nonparametric setting. These distance based methods come with strong statistical guarantees which carry over to our nonparametric generalization. After the structure is recovered, we further use the covariance operator and its principal singular vectors to design surrogates for parameters of the latent variables (called a "spectral algorithm"). One advantage of our spectral algorithm is that it is local-minimum-free and hence amenable for further statistical analysis (see [11, 25, 16] for previous work on spectral algorithms). Last, we will demonstrate the advantage of our method over existing approaches in both simulation and real data experiments.

## 2 Latent Tree Graphical Models

We will focus on latent variable models where the observed variables are continuous and non-Gaussian and the conditional independence structures are specified by trees. We will use uppercase letters to denote random variables (*e.g.*, $X_i$) and lowercase letters their instantiations (*e.g.*, $x_i$). A latent tree model defines a joint distribution over a set, $\mathscr{O} = \{X_1, \ldots, X_O\}$, of $O$ observed variables and a set, $\mathscr{H} = \{X_{O+1}, \ldots, X_{O+H}\}$, of $H$ hidden variables. The complete set of variables is denoted by $\mathscr{X} = \mathscr{O} \cup \mathscr{H}$. For simplicity, we will assume that all observed variables have the same domain $\mathcal{X}_{\mathscr{O}}$, and all hidden variables take values from $\mathcal{X}_{\mathscr{H}}$ and have finite dimension $d$.

The joint distribution of $\mathscr{X}$ in a latent tree model is fully characterized by a set of conditional distributions (CD). More specifically, we can select an arbitrary latent node in the tree as the root, and reorient all edges away from the root. Then the set of CDs between nodes and their parents $\mathbb{P}(X_i|X_{\pi_i})$ are sufficient to characterize the joint distribution (for the root node $X_r$, we set $\mathbb{P}(X_r|X_{\pi_r}) = \mathbb{P}(X_r)$; and we use $\mathbb{P}$ to refer to density in continuous case), $\mathbb{P}(\mathscr{X}) = \prod_{i=1}^{O+H} \mathbb{P}(X_i|X_{\pi_i})$. Compared to tree models which are defined solely on observed variables, latent tree models encompass a much larger classes of models, allowing more flexibility in modeling observed variables. This is evident if we sum out the latent variables in the joint distribution,

$$\mathbb{P}(\mathscr{O}) = \sum_{\mathscr{H}} \prod_{i=1}^{O+H} \mathbb{P}(X_i|X_{\pi_i}). \tag{1}$$

This expression leads to complicated conditional independence structures between observed variables depending on the tree topology. In other words, latent tree models allow complex distributions over observed variables (*e.g.*, clique models) to be expressed in terms of more tractable joint models over the augmented variable space. This can lead to a significant saving in model parametrization.

For simplicity of explanation, we will focus on latent tree structures where each internal node has exactly 3 neighbors. We can reroot the tree and redirect all the edges away from the root. For a variable $X_s$, we use $\alpha_s$ to denote its sibling, $\pi_s$ to denote its parent, $\iota_s$ to denote its left child and $\rho_s$ to denote its right child; the root node will have 3 children, and we use $\omega_s$ to denote the extra child. All the observed variables are leaves in the tree, and we will use $\iota_s^*, \rho_s^*, \pi_s^*$ to denote an observed variable which is found by tracing in the direction from node $s$ to its left child $\iota_s$, right child $\rho_s$, and its parent $\pi_s$ respectively. $s^*$ denotes any observed variable in the subtree rooted at node $s$.

## 3  Kernel Density Estimator and Hilbert Space Embedding

Kernel density estimation (KDE) is a nonparametric way of fitting the density of continuous random variables with non-Gaussian statistical features such as multi-modality and skewness [22]. However, traditional KDE cannot model the latent tree structure. In this paper, we will show that the kernel density estimator can be augmented to deal with latent tree structures using a recent concept called Hilbert space embedding of distributions [23]. Next, we will first explain the basic idea of KDE and distribution embeddings, and show how they are related.

**Kernel density estimator.**  Given a set of *i.i.d.* samples $\mathscr{S} = \left\{(x_1^i, \ldots, x_O^i)\right\}_{i=1}^n$ from $\mathbb{P}(X_1, \ldots, X_O)$, KDE estimates the density via

$$\widehat{\mathbb{P}}(x_1, \ldots, x_O) = \frac{1}{n} \sum\nolimits_{i=1}^n \prod\nolimits_{j=1}^O k(x_j, x_j^i), \tag{2}$$

where $k(x, x')$ is a kernel function. A commonly used kernel function, which we will focus on, is the Gaussian RBF kernel $k(x, x') = \frac{1}{\sqrt{2\pi}\sigma} \exp(-\|x - x'\|^2/2\sigma^2)$. For Gaussian RBF kernel, there exists a feature map $\phi : \mathbb{R} \mapsto \mathcal{F}$ such that $k(x, x') = \langle \phi(x), \phi(x') \rangle_{\mathcal{F}}$, and the feature space has the reproducing property, *i.e.* for all $f \in \mathcal{F}$, $f(x) = \langle f, \phi(x) \rangle_{\mathcal{F}}$. Products of kernels are also kernels, which allow us to write $\prod_{j=1}^O k(x_j, x_j')$ as a single inner product $\langle \otimes_{j=1}^O \phi(x_j), \otimes_{j=1}^O \phi(x_j') \rangle_{\mathcal{F}^O}$. Here $\otimes_{j=1}^O \star$ denotes the tensor product of $O$ feature vectors which results in a rank-1 tensor of order $O$. This inner product can be understood by analogy to the finite dimensional case: given $x, y, z, x', y', z' \in \mathbb{R}^d$, $(x^\top x')(y^\top y')(z^\top z') = \langle x \otimes y \otimes z, \; x' \otimes y' \otimes z' \rangle_{\mathbb{R}^{d^3}}$.

**Hilbert space embedding.**  $\mathcal{C}_{\mathscr{O}} := \mathbb{E}_{\mathscr{O}}\left[\otimes_{j=1}^O \phi(X_j)\right]$ is called the Hilbert space embedding of distribution $\mathbb{P}(\mathscr{O})$ with tensor features $\otimes_{j=1}^O \phi(X_j)$. In other words, the embedding of a distribution is simply the expected feature of that distribution. The essence of Hilbert space embedding is to represent distributions as elements in Hilbert spaces, and then subsequent manipulation of the distributions can be carried out via Hilbert space operations such as inner product and distance. We next show how to represent a KDE using distribution embeddings.

Taking the expected value of a KDE with respect to the random sample $\mathscr{S}$,

$$\mathbb{E}_{\mathscr{S}}\left[\widehat{\mathbb{P}}(x_1, \ldots, x_O)\right] = \mathbb{E}_{\mathscr{O}}\left[\prod\nolimits_{j=1}^O k(x_j, X_j)\right] = \left\langle \mathbb{E}_{\mathscr{O}}\left[\otimes_{j=1}^O \phi(X_j)\right], \otimes_{j=1}^O \phi(x_j) \right\rangle_{\mathcal{F}^O}, \tag{3}$$

we see that this expected value is the inner product between the embedding $\mathcal{C}_{\mathscr{O}}$ and tensor features $\otimes_{j=1}^O \phi(x_j)$. If we replace the embedding $\mathcal{C}_{\mathscr{O}}$ by its finite sample estimate $\widehat{\mathcal{C}}_{\mathscr{O}} := \frac{1}{n} \sum_{i=1}^n \left(\otimes_{j=1}^O \phi(x_j^i)\right)$, we recover the density estimator in (2). Alternatively, using tensor notation (described in supplemental), we can rewrite equation (3) as

$$\left\langle \mathbb{E}_{\mathscr{O}}\left[\otimes_{j=1}^O \phi(X_j)\right], \otimes_{j=1}^O \phi(x_j) \right\rangle_{\mathcal{F}^O} = \mathcal{C}_{\mathscr{O}} \bar{\times}_O \phi(x_O) \ldots \bar{\times}_2 \phi(x_2) \bar{\times}_1 \phi(x_1) \tag{4}$$

where $\mathcal{C}_{\mathscr{O}}$ is a big tensor of order $O$ which can be difficult to store and maintain. While traditional KDE can not make use of the fact that the embedding $\mathcal{C}_{\mathscr{O}}$ originates from a distribution with latent tree structure, the embedding view actually allows us to exploit this special structure and further decompose $\mathcal{C}_{\mathscr{O}}$ to simpler tensors of much lower orders.

## 4  Kernel Embedding of Latent Tree Graphical Models

In this section, we assume that the structures of the latent tree graphical models are *given*, and we will deal with structure learning in the next section. We will show that the tensor expression of KDE in (4) can be computed recursively using a collection of lower order tensors. Essentially, these lower order tensors correspond to the conditional densities in the latent tree graphical models; and the recursive computations try to integrate out the latent variables in the model, and they correspond to the steps in the message passing algorithm for graphical model inference. The challenge is that message passing algorithm becomes nontrivial to represent and implement in continuous and nonparametric settings. Previous methods may lead to exponential blowup in their message representation and hence various approximations are needed, such as expectation propagation [15], mixture of Gaussian simplification [27], and sampling [12]. In contrast, the distribution embedding view allows us to represent and implement message passing algorithm efficiently without resorting to approximations. Furthermore, it also allows us to develop a local-minimum-free algorithm for learning the parameters of latent tree graphical models.

## 4.1 Covariance Operator and Conditional Embedding Operator

We will first explain the concept of conditional embedding operators which are the nonparametric counterparts for conditional probability tables in the discrete case. Conditional embedding operators will be the key building blocks to a nonparametric message passing algorithm as much as conditional probability tables are to the ordinary message passing algorithm.

Following [7], we first define the covariance operator $\mathcal{C}_{X_s X_t}$ which allows us to compute the expectation of the product of function $f(X_s)$ and $g(X_t)$, *i.e.*, $\mathbb{E}_{X_s X_t}[f(X_s)g(X_t)]$, using linear operations in the RKHS. More formally, let $\mathcal{C}_{X_s X_t} : \mathcal{F} \mapsto \mathcal{F}$ such that for all $f, g \in \mathcal{F}$,

$$\mathbb{E}_{X_s X_t}[f(X_s)g(X_t)] = \langle f, \ \mathbb{E}_{X_s X_t}[\phi(X_s) \otimes \phi(X_s)] \, g \rangle_{\mathcal{F}} = \langle f, \ \mathcal{C}_{X_s X_t} g \rangle_{\mathcal{F}} = \mathcal{C}_{st} \ \bar{\times}_2 \ g \ \bar{\times}_1 \ f \quad (5)$$

where we abbreviate the notation $\mathcal{C}_{X_s X_t}$ as $\mathcal{C}_{st}$, and will follow such abbreviation in the rest of the paper (e.g. $\mathcal{C}_{s^2}$ is an abbreviation for $\mathcal{C}_{X_s X_s}$). This can be understood by analogy with the finite dimensional case: if $x, y, z, v \in \mathbb{R}^d$, then $x^\top (yz^\top)v = (yz^\top) \ \bar{\times}_2 \ v \ \bar{\times}_1 \ x$ where we use the tensor-vector multiplication notation from [13] (see supplemental for details). In other words, the covariance operator is also the embedding of the joint distribution $\mathbb{P}(X_s, X_t)$.

Then the conditional embedding operator can be defined via covariance operators according to Song et al. [26]. A conditional embedding operator allows us to compute conditional expectations $\mathbb{E}_{X_t|x_s}[f(X_t)]$ as linear operations in the RKHS. Let $\mathcal{C}_{t|s} := \mathcal{C}_{ts} \mathcal{C}_{ss}^{-1}$ such that for all $f \in \mathcal{F}$,

$$\mathbb{E}_{X_t|x_s}[f(X_t)] = \ \langle f, \ \mathbb{E}_{X_t|x_s}[\phi(X_t)] \rangle_{\mathcal{F}} = \ \langle f, \ \mathcal{C}_{t|s}\phi(x_s) \rangle_{\mathcal{F}} = \mathcal{C}_{t|s} \ \bar{\times}_2 \ \phi(x_s) \ \bar{\times}_1 \ f. \quad (6)$$

In other words, the operator $\mathcal{C}_{t|s}$ takes the feature map $\phi(x_s)$ of the point on which we condition, and outputs the conditional expectation of the feature $\phi(X_t)$ with respect to $\mathbb{P}(X_t|x_s)$. Although the formula looks similar to the Gaussian case, it is important to note that the conditional embedding operator allows us to compute the conditional expectation of *any* $f \in \mathcal{F}$, regardless of the distribution of the random variable in feature space (aside from the condition that $h(\cdot) := \mathbb{E}_{X_t|X_s=\cdot}[f(X_t)]$ is in the RKHS on $X_s$, as noted by Song et al.). In particular, we do not need to assume the random variables have a Gaussian distribution in feature space.

## 4.2 Representation for Message Passing Algorithm

For simplicity, we will focus on latent trees where all latent variables have degree 3 (but our method can be generalized to higher degrees). We first introduce latent variables into equation (3), $\mathbb{E}_{\mathcal{O} \cup \mathcal{H}} \left[ \prod_{j=1}^{O} k(x_j, X_j) \right]$; Then we integrate out the latent variables according to the latent tree structure using a message passing algorithm [17],

* At a leaf node (always observed variable) we pass the following message to its parent $m_s(X_{\pi_s}) = \mathbb{E}_{X_s|X_{\pi_s}}[k(x_s, X_s)]$.

** An internal latent variable aggregates incoming messages from its two children and then sends an outgoing message to its own parent $m_s(X_{\pi_s}) = \mathbb{E}_{X_s|X_{\pi_s}}[m_{\iota_s}(X_s)m_{\rho_s}(X_s)]$.

*** Finally, at the root node, all incoming messages are multiplied together and the root variable is integrated out $b_r := \mathbb{E}_{\mathcal{O}}[\prod_{j=1}^{O} k(x_j, X_j)] = \mathbb{E}_{X_r}[m_{\iota_s}(X_r)m_{\rho_s}(X_r)m_{\omega_r}(X_r)]$.

The challenge is that message passing becomes nontrivial to represent and implement in continuous and nonparametric settings. Previous methods may lead to exponential blowup in their message representation and hence various approximations are needed, such as expectation propagation [15], mixture of Gaussian simplification [27], and sampling [12].

Song et al. [24] show that the above 3 message update operations can be expressed using Hilbert space embeddings [26], and no further approximation is needed in the message computation. Basically, the embedding approach assume that messages are functions in the reproducing kernel Hilbert space, and message update is an operator that takes several functions as inputs and output another function in the reproducing kernel Hilbert space. More specifically, message updates are linear (or multi-linear) operations in feature space,

* At leaf nodes, we have $m_{ts}(\cdot) = \mathbb{E}_{X_s|X_{\pi_s}=\cdot}[k(x_s, X_s)] = \mathcal{C}_{s|\pi_s}^\top \phi(x_s) = \mathcal{C}_{s|\pi_s} \ \bar{\times}_1 \ \phi(x_s)$

** At internal nodes, we define a tensor product reproducing kernel Hilbert space $\mathcal{H} := \mathcal{F} \otimes \mathcal{F}$, under which the product of incoming messages can be written as a single inner product,
$$m_{\iota_s}(X_s) \, m_{\rho_s}(X_s) = \langle m_{\iota_s}, \phi(X_s) \rangle \langle m_{\rho_s}, \phi(X_s) \rangle = \langle m_{\iota_s} \otimes m_{\rho_s}, \phi(X_s) \otimes \phi(X_s) \rangle_{\mathcal{H}}$$
Then the message update becomes
$$m_s(\cdot) = \left\langle m_{\iota_s} \otimes m_{\rho_s}, \ \mathbb{E}_{X_s|X_{\pi_s}=\cdot}[\phi(X_s) \otimes \phi(X_s)] \right\rangle_{\mathcal{H}} = \mathcal{C}_{s^2|\pi_s} \ \bar{\times}_2 \ m_{\rho_s} \ \bar{\times}_1 \ m_{\iota_s} \quad (7)$$

where we define the conditional embedding operator for the tensor features $\phi(X_s) \otimes \phi(X_s)$. By analogy with (6)), $\mathcal{C}_{s^2|\pi_s}$ is defined in terms of a covariance operator $\mathcal{C}_{s^2\pi_s} := \mathbb{E}_{X_s X_{\pi_s}}[\phi(X_s) \otimes \phi(X_s) \otimes \phi(X_s)]$, and the operator $\mathcal{C}_{\pi_s \pi_s}^{-1}$.

*** Finally, at the root nodes, we use the property of tensor product features and arrives at:

$$\mathbb{E}_r[m_{\iota_r}(X_r)\, m_{\rho_r}(X_r)\, m_{\omega_r}(X_r)] = \langle m_{\iota_r} \otimes m_{\rho_r} \otimes m_{\omega_r}, \mathbb{E}_{X_r}[\phi(X_r) \otimes \phi(X_r) \otimes \phi(X_r)]\rangle$$
$$= \mathcal{C}_{r^3} \times_3 m_{\omega_r} \times_2 m_{\rho_r} \times_1 m_{\iota_r} \tag{8}$$

We note that the traditional kernel density estimator needs to estimate a tensor of order $O$ involving all observed variables (equation (4)). By making use of the conditional independence structure of latent tree models, we only need to estimate tensors of much smaller orders. Particularly, we only need to estimate tensors involving two variables (for each parent-child pair), and then the density can be estimated via message passing algorithms using these tensors of much smaller order.

The drawback of the representations in (7) and (8) is that they require exact knowledge of conditional embedding operators associated with latent variables, but none of these are available in training. Next we will show that we can still make use of the tensor decomposition representation without the need for recovering the latent variables explicitly.

### 4.3 Spectral Algorithm for Learning Latent Tree Parameters

Our observation from (7) and (8) is that if we can recover the conditional embedding operators associated with latent variables up to some *invertible* transformations, we will still be able to compute latent tree density correctly. For example, we can transform the messages: $\widetilde{m}_{\iota_s} = T_{\iota_s} m_{\iota_s}$, $\widetilde{m}_{\rho_s} = T_{\rho_s} m_{\rho_s}$, and $\widetilde{m}_{\omega_s} = T_{\omega_s} m_{\omega_s}$, and we can update these transformed messages:

* At leaf nodes, $\widetilde{m}_s = T_s^\top C_{s|\pi_s}^\top \phi(x_s)$

** At internal nodes, $\widetilde{m}_s = (\mathcal{C}_{s^2|\pi_s} \times_1 T_{\iota_s}^{-1} \times_2 T_{\rho_s}^{-1} \times_3 T_s^\top) \bar{\times}_2 \widetilde{m}_{\rho_s} \bar{\times}_1 \widetilde{m}_{\iota_s}$

*** At the root, $b_r = (\mathcal{C}_{r^3} \times_1 T_{\iota_r}^{-1} \times_2 T_{\rho_r}^{-1} \times_3 T_{\omega_r}^{-1}) \bar{\times}_3 \widetilde{m}_{\omega_r} \bar{\times}_2 \widetilde{m}_{\rho_r} \bar{\times}_1 \widetilde{m}_{\iota_r}$

without changing the final $b_r$. Basically, all the invertible transformations $T$ cancel out with each other. These transformations provide us an additional degree of freedom for algorithm design: we can choose the invertible transforms cleverly, such that the transformed representation can be recovered from observed quantities without the need for accessing the latent variables. This representation is related to but different from that of [16] for discrete variables which uses only 3rd order tensors. The kernel case is more challenging and requires $q^{th}$ order tensors (where $q$ is the degree of a node).

More specifically, these transformations $T$ can be constructed from cross covariance operators of certain pairs of observed variables and their singular vectors $U$. We set $T_s = (U_s^\top \mathcal{C}_{s^*|\pi_s})^{-1}$ and let $U_s$ be the top $d$ right eigenvectors of $\mathcal{C}_{\pi_s^* s^*}$. Consider the simple case for the leaf node (∗). In this case, we can set $s^* = s$ and get that $T_s^{-1} = U_s^\top \mathcal{C}_{s|\pi_s}$. Consider the following expansion:

$$\widetilde{m}_s^\top (U_s^\top \mathcal{C}_{s\pi_s^*}) = \phi^T(x_s) C_{s|\pi_s} (U_s^\top C_{s|\pi_s})^{-1} (U_s^\top C_{s|\pi_s})(\mathcal{C}_{\pi_s^2} \mathcal{C}_{\pi_s^*|\pi_s}^\top) = \phi(x_s)^\top C_{s\pi_s^*} \tag{9}$$
$$\Rightarrow \quad \widetilde{m}_s = (\mathcal{C}_{\pi_s^* s} U_s)^\dagger C_{\pi_s^* s} \phi(x_s) \tag{10}$$

Here † denotes pseudo-inverse. The general pattern is that we can relate the transformed latent quantity to observed quantities in two different ways such that we can solve for the transformed latent quantity. A similar strategy can be applied to $\widetilde{\mathcal{C}}_{s^2|\pi_s} := \mathcal{C}_{s^2|\pi_s} \times_1 T_{\iota_s}^{-1} \times_2 T_{\rho_s}^{-1} \times_3 T_s^\top$ in the internal message update, and the $\widetilde{\mathcal{C}}_{r^3} := \mathcal{C}_{r^3} \times_1 T_{\iota_s}^{-1} \times_2 T_{\rho_s}^{-1} \times_3 T_{\omega_r}^{-1}$ at the root. We summarize the results on how to compute the transformed quantities below (see supplemental for details).

* At leaf nodes, $\widetilde{m}_s = (\mathcal{C}_{\pi_s^* s} U_s)^\dagger C_{\pi_s^* s} \phi(x_s)$.

** At internal nodes, $\widetilde{\mathcal{C}}_{s^2|\pi_s} = \mathcal{C}_{\iota_s^* \rho_s^* \pi_s^*} \times_1 U_{\iota_s}^\top \times_2 U_{\rho_s}^\top \times_3 (\mathcal{C}_{\pi_s^* \iota_s^*} U_s)^\dagger$.

*** At the root, $\widetilde{\mathcal{C}}_{r^3} = \mathcal{C}_{\iota_r^* \rho_r^* \omega_r^*} \times_1 U_{\iota_r}^\top \times_2 U_{\rho_r}^\top \times_3 U_{\omega_r}^\top$.

The above results give us an efficient algorithm for computing the expected kernel density $b_r$ which can take into account the latent tree structures while at the same time avoiding the local minimum problems associated with explicitly recovering latent parameters. The main computation only involves tensor-matrix and tensor-vector multiplications, and a sequence of singular value decompositions of pairwise cross covariance operators. After we obtain the transformed quantities, we can then use them in the message passing algorithm to obtain the final belief $b_r$.

Given a sample $\mathscr{S} = \{(x_1^i, \ldots, x_O^i)\}_{i=1}^n$ drawn *i.i.d.* from a $\mathbb{P}(\mathscr{O})$, the spectral algorithm for latent trees proceeds by replacing all population quantities by their empirical counterpart. For instance,

the SVD of covariance operators between $X_s$ and $X_t$ can be estimated by first forming matrices $\Upsilon = (\phi(x_s^1), \ldots, \phi(x_s^n))$ and $\Phi = (\phi(x_t^1), \ldots, \phi(x_t^n))$, and estimate $\widehat{\mathcal{C}}_{ts} = \frac{1}{n}\Phi\Upsilon^\top$. Then a singular value decomposition of $\widehat{\mathcal{C}}$ can be carried out to obtain an estimate for $\widehat{U}$ (See [25] for more details).

# 5   Structure Learning of Latent Tree Graphical Models

The last section focused on density estimation where the structure of the latent tree is known. In this section, we focus on learning the structure of the latent tree. Structure learning of latent trees is a challenging problem that has largely been tackled by heuristics since the search space of structures is intractable. The additional challenge in our case is that the observed variables are continuous and non-Gaussian, which we are not aware of any existing methods for this problem.

**Structure learning algorithm** We develop a distance based method for constructing latent trees of continuous, non-Gaussian variables. The idea is that if we have a tree metric (distance) between distributions on observed nodes, we can use the property of the tree metric to reconstruct the latent tree structure using algorithms such as neighbor joining [20] and the recursive grouping algorithm [4]. These methods take a distance matrix among all pairs of observed variables as input and output a tree by iteratively adding hidden nodes. While these methods are iterative, they have strong theoretical guarantees on structure recovery when the true distance matrix forms an additive tree metric. However, most previously known tree metrics are defined for discrete and Gaussian variables. The additional challenge in our case is that the observed variables are continuous and non-Gaussian. We propose a tree metric below which works for continuous non-Gaussian cases.

**Tree metric and pseudo-determinant** We will first explain some basic concepts of a tree metric. If the joint probability distribution $\mathbb{P}(\mathscr{X})$ has a latent tree structure, then a distance measure $d_{st}$ between an arbitrary variables pairs $X_s$ and $X_t$ are called tree metric if it satisfies the following path additive condition: $d_{st} = \sum_{(u,v)\in Path(s,t)} d_{uv}$. For discrete and Gaussian variables, tree metric can be defined via the determinant $|\cdot|$ [4]

$$d_{st} = -\tfrac{1}{2}\log|C_{st}C_{st}^\top| + \tfrac{1}{4}\log|C_{ss}C_{ss}^\top| + \tfrac{1}{4}\log|C_{tt}C_{tt}^\top|, \tag{11}$$

where $C_{st}$ denotes joint probability matrix in the discrete case and the covariance in the Gaussian case; $C_{ss}$ is the diagonalized marginal probability vector in the discrete case and variance in the Gaussian case. However, this definition of tree metric is restricted in the sense that it requires all discrete variables to have the same number of states and all Gaussian variables have the same dimension. This is because determinant is only defined (and non-zero) for square and non-singular matrices. For our more general scenario, where the observed variables are continuous non-Gaussian but the hidden variables have dimension $d$, we will define a tree metric based on pseudo-determinant which works for our operators.

**Nonparametric tree metric** The pseudo-determinant is defined as the product of non-zero singular values of an operator $|\mathcal{C}|_\star = \prod_{i=1}^d \sigma_i(\mathcal{C})$. In our case, since we assume that the dimension of the hidden variables is $d$, the pseudo-determinant is simply the product of top $d$ singular values. Then we define the distance metric between two continuous non-Gaussian variables $X_s$ and $X_t$ as

$$d_{st} = -\tfrac{1}{2}\log\left|\mathcal{C}_{st}\mathcal{C}_{st}^\top\right|_\star + \tfrac{1}{4}\log|\mathcal{C}_{ss}\mathcal{C}_{ss}^\top|_\star + \tfrac{1}{4}\log|\mathcal{C}_{tt}\mathcal{C}_{tt}^\top|_\star. \tag{12}$$

One can prove that (12) defines a tree metric by inducting on the path length. Here we only show the additive property for the simplest path $X_s - X_u - X_t$ involving only a single hidden variable $X_u$. In this case, we first factorize $|\mathcal{C}_{st}\mathcal{C}_{st}^\top|_\star$ into $|\mathcal{C}_{s|u}\mathcal{C}_{uu}\mathcal{C}_{t|u}^\top\mathcal{C}_{t|u}\mathcal{C}_{uu}\mathcal{C}_{s|u}^\top|_\star$ according to the Markov property. Then using Sylvester's determinant theorem, the latter is also equal to $|\mathcal{C}_{s|u}^\top\mathcal{C}_{s|u}\mathcal{C}_{uu}\mathcal{C}_{t|u}^\top\mathcal{C}_{t|u}\mathcal{C}_{uu}|_\star$ by flipping $\mathcal{C}_{s|u}^\top$ to the front. Next, introducing two copies of $|\mathcal{C}_{uu}|$ and rearranging terms, we have

$$|\mathcal{C}_{st}\mathcal{C}_{st}^\top|_\star = \frac{|\mathcal{C}_{s|u}\mathcal{C}_{uu}\mathcal{C}_{uu}\mathcal{C}_{s|u}^\top|_\star|\mathcal{C}_{t|u}\mathcal{C}_{uu}\mathcal{C}_{uu}\mathcal{C}_{t|u}^\top|_\star}{|\mathcal{C}_{uu}|_\star|\mathcal{C}_{uu}|_\star} = \frac{|\mathcal{C}_{su}\mathcal{C}_{su}^\top|_\star|\mathcal{C}_{tu}\mathcal{C}_{tu}^\top|_\star}{|\mathcal{C}_{uu}\mathcal{C}_{uu}|_\star}. \tag{13}$$

Last, we plug this into (12) and we have the desired path additive property

$$\begin{aligned} d_{st} &= -\tfrac{1}{2}\log|\mathcal{C}_{su}\mathcal{C}_{su}^\top|_\star - \tfrac{1}{2}\log|\mathcal{C}_{tu}\mathcal{C}_{tu}^\top|_\star + \tfrac{1}{2}\log|\mathcal{C}_{uu}\mathcal{C}_{uu}^\top|_\star + \tfrac{1}{4}\log|\mathcal{C}_{ss}\mathcal{C}_{ss}^\top|_\star + \tfrac{1}{4}\log|\mathcal{C}_{tt}\mathcal{C}_{tt}^\top| \\ &= d_{su} + d_{ut} \end{aligned}$$

# 6   Experiments

We evaluate our method on synthetic data as well as a real-world crime/communities dataset [1, 19]. For all experiments we compare to 2 existing approaches. The first is to assume the data are

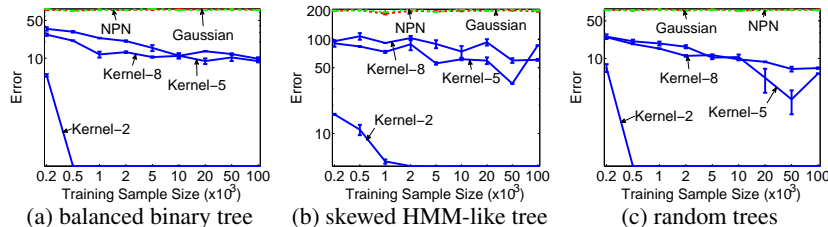

(a) balanced binary tree     (b) skewed HMM-like tree     (c) random trees

Figure 1: Comparison of our kernel structure learning method to the Gaussian and Nonparanormal methods on different tree structures.

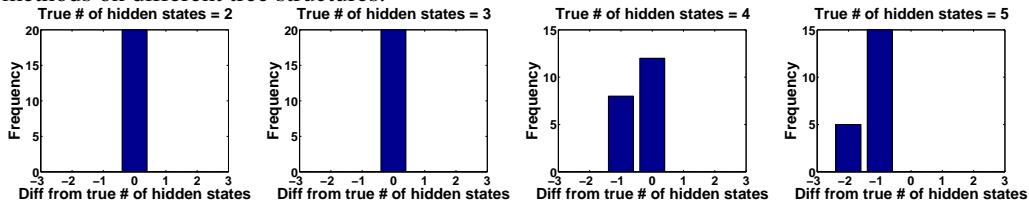

Figure 2: Histogram of the differences between the estimated number of hidden states and the true number of states.

multivariate Gaussians and use the tree metric defined in [4] (which is essentially a function of the correlation coefficient). The second existing approach we compare to is the Nonparanormal (NPN) [14] which assumes that there exist marginal transformations $f_1, \ldots, f_p$ such that $f(X_1), \ldots, f(X_p) \sim N(\mu, \Sigma)$. If the data comes from a Nonparanormal distribution, then the transformed data are assumed to be multivariate Gaussians and the same tree metric as the Gaussian case can be used on the transformed data. Our approach makes much fewer assumptions about the data than either of these two methods which can be more favorably in practice.

To perform learning and inference in our approach, we use the spectral algorithm and message passing algorithm described earlier in the paper. For inference in the Gaussian (and nonparanormal) cases, we use the technique in [4] to learn the model parameters (covariance matrix). Once the covariance matrix has been estimated, computing the marginal of one variable given a set of evidence reduces to solving a linear equation of one variable [2].

**Synthetic data: structure recovery.** The first experiment is to demonstrate how our method compares to the Gaussian and Nonparanormal methods in terms of structure recovery. We experiment with 3 different tree types (each with 64 leaves or observed variables): a balanced binary tree, a completely binary skewed tree (like an HMM), and randomly generated binary trees. For all trees, we use the following generative process to generate the $n$-th sample from a node $s$ (denoted $x_s^{(n)}$): If $s$ is the root, sample from a mixture of 2 Gaussians. Else, with probability $\frac{1}{2}$ sample from a Gaussian with mean $-x_{\pi_s}^{(n)}$ and with probability $\frac{1}{2}$ sample from a Gaussian with mean $x_{\pi_s}^{(n)}$.

We vary the training sample size from 200 to 100,000. Once we have computed the empirical tree distance matrix for each algorithm, we use the neighbor joining algorithm [20] to learn the trees. For evaluation we compare the number of hops between each pair of leaves in the true tree to the estimated tree. For a pair of leaves $i, j$ the error is defined as: $error(i,j) = \frac{|hops^*(i,j) - \widehat{hops}(i,j)|}{hops^*(i,j)} + \frac{|hops^*(i,j) - \widehat{hops}(i,j)|}{\widehat{hops}(i,j)}$, where $hops^*$ is the true number of hops and $\widehat{hops}$ is the estimated number of hops. The total error is then computed by adding the error for each pair of leaves.

The performance of our method depends on the number of singular values chosen and we experimented with 2, 5 and 8 singular values. Furthermore, we choose the bandwidth $\sigma$ for the Gaussian RBF kernel needed for the covariance operators using median distance between pairs of training points. For all these choices our method performs better than the Gaussian and Nonparanormal methods. This is to be expected, since the data we generated is neither Gaussian or Nonparamnormal, yet our method is able to learn the structure correctly. We also note that balanced binary trees are the easiest to learn while the skewed trees are the hardest (Figure 1).

**Synthetic data: model selection.** Next we evaluate the ability of our model to select the correct number of singular values via held-out likelihood. For this experiment we use a balanced binary tree with 16 leaves (total of 31 nodes) and 100000 samples. A different generative process is used so

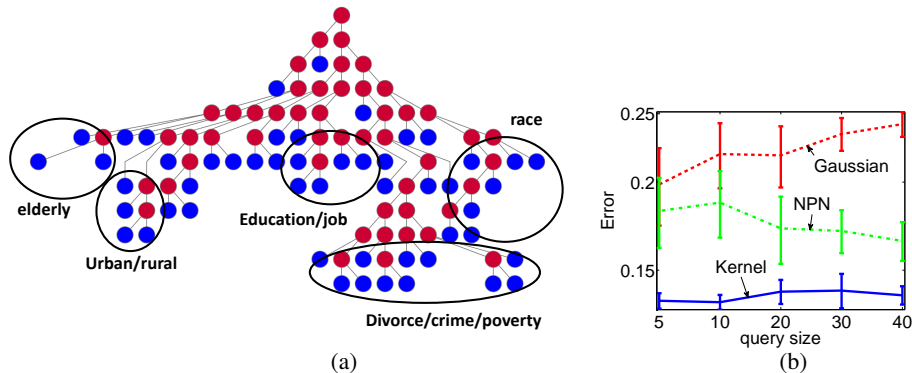

Figure 3: (a) visualization of kernel latent tree learned from crime data (b) Comparison of our method to Gaussian and NPN in predictive task.

that it is clear what the correct number of singular values should be (When the hidden state space is continuous like in our first synthetic experiment this is unclear). Each internal node is discrete and takes on $d$ values. The leaf is a mixture of $d$ Gaussians where which Gaussian to sample from is dictated by the discrete value of the parent.

We vary $d$ from 2 through 5 and then run our method for a range of 2 through 8 singular values. We select the model that has the highest likelihood computed using our spectral algorithm on a hold-out set of 500 examples. We then take the difference between the number of singular values chosen and the true singular values, and plot histograms of this difference (Ideally all the trials should be in the zero bin). The experiment is run for 20 trials. As we can see in Figure 2, when $d$ is low, the held-out likelihood computed by our method does a fairly good job in recovering the correct number. However, as the true number of eigenvalues rises our method underestimates the true number (although it is still fairly close).

**Crime Data.** Finally, we explore the performance of our method on a communities and crime dataset from the UCI repository [1, 19]. In this dataset several real valued attributes are collected for several communities, such as ethnicity proportions, income, poverty rate, divorce rate etc., and the goal is to predict the number of violent crimes (proportional to the size of the community) that occur based on these attributes. In general these attributes are highly skewed and therefore not well characterized by a Gaussian model.

We divide the data into 1400 samples for training, 300 samples for model selection (held-out likelihood), and 300 samples for testing. We pick the first 50 of these attributes, plus the violent crime variable and construct a latent tree using our tree metric and the neighbor joining algorithm [20]. We depict the tree in Figure 3 and highlight a few coherent groupings. For example, the "elderly" group attributes are those related to retirement and social security (and thus correlated). The large clustering in the center is where the class variable (violent crimes) is located next to the poverty rate and the divorce rate among other relevant variables. Other groupings include type of occupation and education level as well as ethnic proportions. Thus, overall our method captures sensible relationships.

For a more quantitative evaluation, we condition on a set of $\mathcal{E}$ of evidence variables and predict the violent crimes class label. We experiment with a varying number of sizes of evidence sets from 5 to 40 and repeat for 40 randomly chosen evidence sets of a fixed size. Since the crime variable is a number between 0 and 1, our error measure is simply $err(\hat{c}) = |\hat{c} - c^*|$ (where $\hat{c}$ is the predicted value and $c^*$ is the true value. As one can see in Figure 3 our method outperforms both the Gaussian and the nonparanormal for the range of query sizes. Thus, in this case our method is better able to capture the skewed distributions of the variables than the other methods.

### Acknowledgments

This work was partially done when LS was at Carnegie Mellon University and Google Research. This work is also supported by an NSF Graduate Research Fellowship (under Grant No. 0750271) to APP, NIH 1R01GM093156, NIH 1RC2HL101487, NSF DBI-0546594, and an Alfred P. Sloan Fellowship to EPX.

# References

[1] A. Asuncion and D.J. Newman. Uci machine learning repository, 2007.

[2] D. Bickson. Gaussian belief propagation: Theory and application. *Arxiv preprint arXiv:0811.2518*, 2008.

[3] D. Blei, A. Ng, and M. Jordan. Latent dirichlet allocation. In *NIPS*, 2002.

[4] M.J. Choi, V.Y.F. Tan, A. Anandkumar, and A.S. Willsky. Learning latent tree graphical models. *Arxiv preprint arXiv:1009.2722*, 2010.

[5] A. Clark. Inference of haplotypes from pcr-amplified samples of diploid populations. *Molecular Biology and Evolution*, 7(2):111–122, 1990.

[6] A. Dempster, N. Laird, and D. Rubin. Maximum likelihood from incomplete data via the em algorithm. *Journal of the Royal Statistical Society B*, 39(1):1–22, 1977.

[7] K. Fukumizu, F. R. Bach, and M. I. Jordan. Dimensionality reduction for supervised learning with reproducing kernel Hilbert spaces. *J. Mach. Learn. Res.*, 5:73–99, 2004.

[8] S. Harmeling and C.K.I. Williams. Greedy learning of binary latent trees. *IEEE Transactions on Pattern Analysis and Machine Intelligence*, 2010.

[9] K.A. Heller and Z. Ghahramani. Bayesian hierarchical clustering. In *Proceedings of the 22nd international conference on Machine learning*, pages 297–304. ACM, 2005.

[10] Peter D. Hoff, Adrian E. Raftery, and Mark S. Handcock. Latent space approaches to social network analysis. *JASA*, 97(460):1090–1098, 2002.

[11] D. Hsu, S. Kakade, and T. Zhang. A spectral algorithm for learning hidden markov models. In *COLT*, 2009.

[12] A. Ihler and D. McAllester. Particle belief propagation. In *AISTATS*, pages 256–263, 2009.

[13] Tamara. Kolda and Brett Bader. Tensor decompositions and applications. *SIAM Review*, 51(3):455–500, 2009.

[14] H. Liu, J. Lafferty, and L. Wasserman. The nonparanormal: Semiparametric estimation of high dimensional undirected graphs. *The Journal of Machine Learning Research*, 10:2295–2328, 2009.

[15] T. Minka. *Expectation Propagation for approximative Bayesian inference*. PhD thesis, MIT Media Labs, Cambridge, USA, 2001.

[16] A. Parikh, L. Song, and E. Xing. A spectral algorithm for latent tree graphical models. In *ICML*, 2011.

[17] J. Pearl. *Causality: Models, Reasoning and Inference*. Cambridge University Press, 2001.

[18] L. R. Rabiner and B. H. Juang. An introduction to hidden Markov models. *IEEE ASSP Magazine*, 3(1):4–16, January 1986.

[19] M. Redmond and A. Baveja. A data-driven software tool for enabling cooperative information sharing among police departments. *European Journal of Operational Research*, 141(3):660–678, 2002.

[20] N. Saitou, M. Nei, et al. The neighbor-joining method: a new method for reconstructing phylogenetic trees. *Mol Biol Evol*, 4(4):406–425, 1987.

[21] C. Semple and M.A. Steel. *Phylogenetics*, volume 24. Oxford University Press, USA, 2003.

[22] B. W. Silverman. *Density Estimation for Statistical and Data Analysis*. Monographs on statistics and applied probability. Chapman and Hall, London, 1986.

[23] A.J. Smola, A. Gretton, L. Song, and B. Schölkopf. A hilbert space embedding for distributions. In E. Takimoto, editor, *Algorithmic Learning Theory*, Lecture Notes on Computer Science. Springer, 2007.

[24] L. Song, A. Gretton, and C. Guestrin. Nonparametric tree graphical models. In *13th Workshop on Artificial Intelligence and Statistics*, volume 9 of *JMLR workshop and conference proceedings*, pages 765–772, 2010.

[25] Le Song, Byron Boots, Sajid Siddiqi, Geoffrey Gordon, and Alex Smola. Hilbert space embeddings of hidden markov models. In *International Conference on Machine Learning*, 2010.

[26] Le Song, Jonathan Huang, Alex Smola, and Kenji Fukumizu. Hilbert space embeddings of conditional distributions. In *ICML*, 2009.

[27] E. Sudderth, A. Ihler, W. Freeman, and A. Willsky. Nonparametric belief propagation. In *CVPR*, 2003.

[28] N.L. Zhang. Hierarchical latent class models for cluster analysis. *The Journal of Machine Learning Research*, 5:697–723, 2004.

